# Phase transitions and the perceptual organization of video sequences

**Yair Weiss**
Dept. of Brain and Cognitive Sciences
Massachusetts Institute of Technology
E10-120, Cambridge, MA 02139
http://www-bcs.mit.edu/~yweiss

## Abstract

Estimating motion in scenes containing multiple moving objects remains a difficult problem in computer vision. A promising approach to this problem involves using mixture models, where the motion of each object is a component in the mixture. However, existing methods typically require specifying in advance the number of components in the mixture, i.e. the number of objects in the scene.

Here we show that the number of objects can be estimated automatically in a maximum likelihood framework, given an assumption about the level of noise in the video sequence. We derive analytical results showing the number of models which maximize the likelihood for a given noise level in a given sequence. We illustrate these results on a real video sequence, showing how the phase transitions correspond to different perceptual organizations of the scene.

Figure 1a depicts a scene where motion estimation is difficult for many computer vision systems. A semi-transparent surface partially occludes a second surface, and the camera is translating horizontally. Figure 1b shows a slice through the horizontal component of the motion generated by the camera - points that are closer to the camera move faster than those further away. In practice, the local motion information would be noisy as shown in figure 1c and this imposes conflicting demands on a motion analysis system - reliable estimates require pooling together many measurements while avoiding mixing together measurements derived from the two different surfaces.

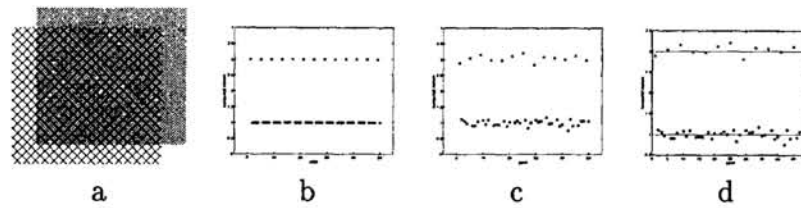

a          b          c          d

Figure 1: **a:** A simple scene that can cause problems for motion estimation. One surface partially occludes another surface. **b:** A cross section through the horizontal motion field generated when the camera translates horizontally. Points closer to the camera move faster. **c:** Noisy motion field. In practice each local measurement will be somewhat noisy and pooling of information is required. **d:** A cross section through the output of a multiple motion analysis system. Points are assigned to surfaces (denoted by different plot symbols) and the motion of each surface is estimated.

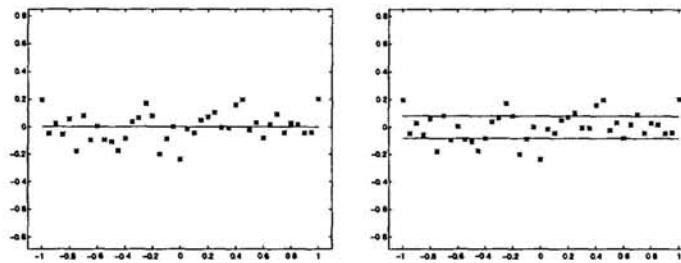

Figure 2: The "correct" number of surfaces in a given scene is often ambiguous. Was the motion here generated by one or two surfaces?

Significant progress in the analysis of such scenes has been achieved by multiple motion analyzers - systems that simultaneously segment the scene into surfaces and estimating the motion of each surface [9]. Mixture models are a commonly used framework for performing multiple motion estimation [5, 1, 10]. Figure 1d shows a slice through the output of a multiple motion analyzer on this scene - pixels are assigned to one of two surfaces and motion information is only combined for pixels belonging to the same surface.

The output shown in figure 1d was obtained by assuming the scene contains two surfaces. In general, of course, one does not know the number of surfaces in the scene in advance. Figure 2 shows the difficulty in estimating this number. It is not clear whether this is very noisy data generated by a single surface, or less noisy data generated by two surfaces. There seems no reason to prefer one description over another. Indeed, the description where there are as many surfaces as pixels is also a valid interpretation of this data.

Here we take the approach that there is no single "correct" number of surfaces for a given scene in the absence of any additional assumptions. However, given an assumption about the noise in the sequence, there are more likely and less likely interpretations. Intuitively, if we know that the data in figure 2a was taken with a very noisy camera, we would tend to prefer the one surface solution - adding additional surfaces would cause us to fit the noise rather than the data. However, if we know that there is little noise in the sequence, we would prefer solutions that use many surfaces, there is a lot less danger of "overfitting". In this paper[1] we show,

following [6, 8] that this intuition regarding the dependence of number of surfaces to assumed noise level is captured in the maximum likelihood framework. We derive analytical results for the critical values of noise levels where the likelihood function undergoes a "phase transition" – from being maximized by a single model to being maximized by multiple models. We illustrate these transitions on synthetic and real video data.

# 1   Theory

## 1.1   Mixture Models for optical flow

In mixture models for optical flow (cf. [5, 1]) the scene is modeled as composed of $K$ surfaces with the velocity of each vsurface at location $(x, y)$ given by $(u^k(x, y), v^k(x, y))$. The velocity field is parameterized by a vector $\theta^k$. A typical choice [9] is the affine representation:

$$u^k(x, y) = \theta_0^k + \theta_1^k x + \theta_2^k y \tag{1}$$
$$v^k(x, y) = \theta_4^k + \theta_5^k x + \theta_6^k y \tag{2}$$

The affine family of motions includes rotations, translations, scalings and shears. It corresponds to the 2D projection of a plane undergoing rigid motion in depth.

Corresponding pixels in subsequent frames are assumed to have identical intensity values, up to imaging noise which is modeled as a Gaussian with variance $\sigma^2$. The task of multiple motion estimation is to find the most likely motion parameter values given the image data. A standard derivation (see e.g. [1]) gives the following log likelihood function for the parameters $\Theta$:

$$l(\Theta) = \sum_{x,y} \log(\sum_{k=1}^{K} e^{-R_k^2(x,y)/2\sigma^2}) \tag{3}$$

With $R_k(x, y)$ the residual intensity at pixel $(x, y)$ for velocity $k$:

$$R_k(x, y) = I_x(x, y)u^k(x, y) + I_y(x, y)v^k(x, y) + I_t(x, y) \tag{4}$$

where $I_x, I_y, I_t$ denote the spatial and temporal derivatives of the image sequence. Although our notation does not make it explicit, $R_k(x, y)$ is a function of $\theta_k$ through equations 1–2. As in most mixture estimation applications, equation 3 is not maximized directly, but rather an Expectation-Maximization (EM) algorithm is used to iteratively increase the likelihood [3].

## 1.2   Maximum Likelihood not necessarily with maximum number of models

It may seem that since $K$ is fixed in the likelihood function (equation 3) there is no way that the number of surfaces can be found by maximizing the likelihood. However, maximizing over the likelihood may lead to a a solution in which some of the $\theta$ parameters are identical [6, 5, 8]. In this case, although the number of surfaces is still $K$, the number of *distinct* surfaces may be any number less than $K$.

Consider a very simple case where $K = 2$ and the motion of each surface is restricted to horizontal translation $u(x, y) = u, v(x, y) = 0$. The advantage of this simplified

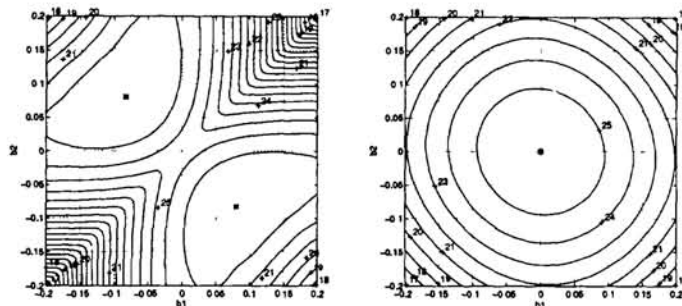

Figure 3: The log likelihood for the data in figure 2 undergoes a phase transition when $\sigma$ is varied. For small values of $\sigma$ the likelihood has two maxima, and at both these maxima the two motions are distinct. For large $\sigma^2$ the likelihood function has single maximum at the origin, corresponding to the solution where both velocities are equal to zero, or only one unique surface.

case is that the likelihood function is a function of two variables and can be easily visualized. Figure 3 shows the likelihood function for the data in figure 2 as $\sigma$ is varied. Observe that for small values of $\sigma^2$ the likelihood has two maxima, and at both these maxima the two motions are distinct. For large $\sigma^2$ the likelihood function has single maximum at the origin, corresponding to the solution where both velocities are equal to zero, or only one unique surface. This is a simple example where the ML solution corresponds to a small number of unique surfaces.

Can we predict the range of values for $\sigma$ for which the likelihood function has a maximum at the origin? This happens when the gradient of the likelihood at the origin is zero and the Hessian has two negative eigenvalues. It is easy to show that the if the data has zero mean, the gradient is zero regardless of $\sigma$. As for the Hessian, $H$, direct calculation gives:

$$H = c \begin{pmatrix} \frac{E}{2\sigma^2} - 1 & -\frac{E}{2\sigma^2} \\ -\frac{E}{2\sigma^2} & \frac{E}{2\sigma^2} - 1 \end{pmatrix} \tag{5}$$

where $E$ is the mean squared residual of a single motion and $c$ is a positive constant. The two eigenvalues are proportional to $-1$ and $E/\sigma^2 - 1$. So the likelihood function has a local maximum at the origin if and only if $E < \sigma^2$. (see [6, 4, 8] for a similar analysis in other contexts).

This result makes intuitive sense. Recall that $\sigma^2$ is the expected noise variance. Thus if the mean squared residual is less than $\sigma^2$ with a single surface, there is no need to add additional surfaces. The result on the Hessian shows that this intuition is captured in the likelihood function. There is no need to introduce additional "complexity costs" to avoid overfitting in this case.

More generally, if we assume the velocity fields are of general parametric form, the Hessian evaluated at the point where both surfaces are identical has the form:

$$H = c \begin{pmatrix} \frac{E}{2\sigma^2} - F & -\frac{E}{2\sigma^2} \\ -\frac{E}{2\sigma^2} & \frac{E}{2\sigma^2} - F \end{pmatrix} \tag{6}$$

where $E$ and $F$ are matrices:

$$E = \sum_{x,y} R^2(x,y) d(x,y) d(x,y)^t \tag{7}$$

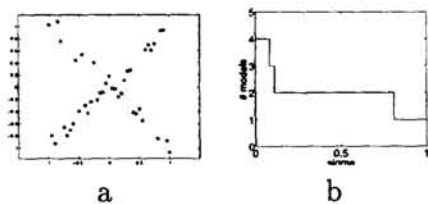

a                                       b

Figure 4: **a:** data generated by two lines. **b:** the predicted phase diagram for the likelihood of this dataset in a four component mixture. The phase transitions are at $\sigma = 0.084, 0.112, 0.8088$

$$F = \sum_{x,y} d(x,y)d(x,y)^t \tag{8}$$

with $d(x,y) = \frac{\partial R(x,y)}{\partial \theta}$, and $R(x,y)$ the residual as before.

A necessary and sufficient condition for the Hessian to have only negative eigenvalues is:

$$\|F^{-1}E\| < \sigma^2 \tag{9}$$

Thus when the maximal eigenvalue of $F^{-1}E$ is less than $\sigma^2$ the fit with a single model is a local maximum of the likelihood. Note that $F^{-1}E$ is very similar to a weighted mean squared error, with every residual weighted by a positive definite matrix (E sums all the residuals times their weight, and F sums all the weights, so $F^{-1}E$ is similar to a weighted average).

The above analysis predicts the phase transition of a two component mixture likelihood, i.e. the critical value of $\sigma^2$ such that above this critical value, the maximum likelihood solution will have identical motion parameters for both surfaces. This analysis can be straightforwardly generalized to finding the first phase transition of a $K$ component mixture, although the subsequent transitions are harder to analyze.

## 2   Results

The fact that the likelihood function undergoes a phase transition as $\sigma$ is varied predicts that a ML technique will converge to different number of distinct models as $\sigma$ is varied. We first illustrate these phase transitions on a $1D$ line fitting problem which shares some of the structure of multiple motion analysis and is easily visualized.

Figure 4a shows data generated by two lines with additive noise, and figure 4b shows a phase diagram calculated using repeated application of equation 9; i.e. by solving equation 9 for all the data, taking the two line solution obtained after the transition, and repeating the calculation separately for points assigned to each of the two lines.

Figure 5 shows the output of an EM algorithm on this data set. Initial conditions are identical in all runs, and the algorithm converges to one, two, three or four distinct lines depending on $\sigma$.

We now illustrate the phase transitions on a real video sequence. Figures 6– 8 show the output of an EM motion segmentation algorithm with four components on the MPEG flower garden sequence (cf. [9, 10]). The camera is translating in

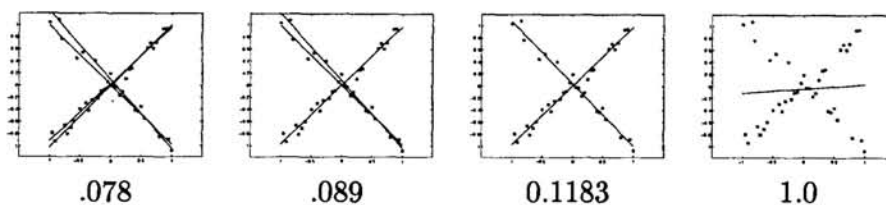

.078　　　　　　.089　　　　　　0.1183　　　　　　1.0

Figure 5: The data in figure 1 are fit with one, two, three or four models depending on $\sigma$. The results of EM with identical initial conditions are shown, only $\sigma$ is varied. The transitions are consistent with the theoretical predictions.

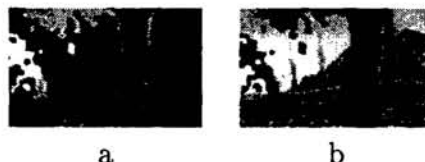

a　　　　　　b

Figure 6: The first phase transition. The algorithm finds two segments corresponding to the tree and the rest of the scene. The critical value of $\sigma^2$ for which this transition happens is consistent with the theoretical prediction.

the scene, and objects move with different velocities due to parallax. The phase transitions correspond to different perceptual organizations of the scene - first the tree is segmented from the background, then branches are split from the tree, and finally the background splits into the flower bed and the house.

## 3　Discussion

Estimating the number of components in a Gaussian mixture is a well researched topic in statistics and data mining [7]. Most approaches involve some tradeoff parameter to balance the benefit of an additional component versus the added complexity [2]. Here we have shown how this tradeoff parameter can be implicitly specified by the assumed level of noise in the image sequence.

While making an assumption regarding $\sigma$ may seem rather arbitrary in the abstract Gaussian mixture problem, we find it quite reasonable in the context of motion estimation, where the noise is often a property of the imaging system, not of the underlying surfaces. Furthermore, as the phase diagram in figure 4 shows, a wide range of assumed $\sigma$ values will give similar answer, suggesting that an exact specification of $\sigma$ is not needed. In current work we are exploring the use of weak priors on $\sigma$ as well as comparing our method to those based on cross validation [7].

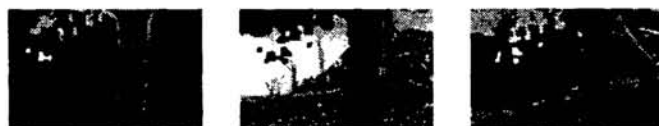

Figure 7: The second phase transition. The algorithm finds three segments - branches which are closer to the camera than the rest of the tree are segmented from it. Since the segmentation is based solely on motion, portions of the flower bed that move consistently with the branches are erroneously grouped with them.

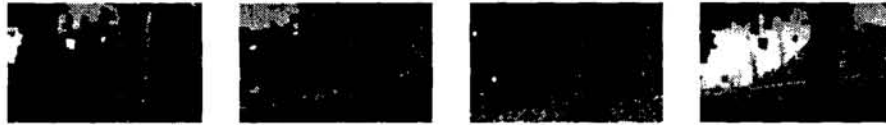

Figure 8: The third phase transition. The algorithm finds four segments – the flower bed and the house are segregated.

Our analytical and simulation results show that an assumption of the noise level in the sequence enables automatic determination of the number of moving objects using well understood maximum likelihood techniques. Furthermore, for a given scene, varying the assumed noise level gives rise to different perceptually meaningful segmentations. Thus mixture models may be a first step towards a well founded probabilistic framework for perceptual organization.

## Acknowledgments

I thank D. Fleet, E. Adelson, J. Tenenbaum and G. Hinton for stimulating discussions. Supported by a training grant from NIGMS.

## Footnotes

[1]A longer version of this paper is available on the author's web page.

# References

[1] Serge Ayer and Harpreet S. Sawhney. Layered representation of motion video using robust maximum likelihood estimation of mixture models and MDL encoding. In *Proc. Int'l Conf. Comput. Vision*, pages 777–784, 1995.

[2] J. Buhmann. Data clustering and learning. In M. Arbib, editor, *Handbook of Brain Theory and Neural Networks*. MIT Press, 1995.

[3] A. P. Dempster, N. M. Laird, and D. B. Rubin. Maximum likelihood from incomplete data via the EM algorithm. *J. R. Statist. Soc. B*, 39:1–38, 1977.

[4] R. Durbin, R. Szeliski, and A. Yuille. An analysis of the elastic net approach to the travelling salesman problem. *Neural Computation*, 1(3):348–358, 1989.

[5] A. Jepson and M. J. Black. Mixture models for optical flow computation. In *Proc. IEEE Conf. Comput. Vision Pattern Recog.*, pages 760–761, New York, June 1993.

[6] K. Rose, F. Gurewitz, and G. Fox. Statistical mechanics and phase transitions in clustering. *Physical Review Letters*, 65:945–948, 1990.

[7] P. Smyth. Clustering using monte-carlo cross-validation. In *KDD-96*, pages 126–133, 1996.

[8] J. B. Tenenbaum and E. V. Todorov. Factorial learning by clustering features. In G. Tesauro, D.S. Touretzky, and K. Leen, editors, *Advances in Neural Information Processing Systems 7*, 1995.

[9] J. Y. A. Wang and E. H. Adelson. Representing moving images with layers. *IEEE Transactions on Image Processing Special Issue: Image Sequence Compression*, 3(5):625–638, September 1994.

[10] Y. Weiss and E. H. Adelson. A unified mixture framework for motion segmentation: incorporating spatial coherence and estimating the number of models. In *Proc. IEEE Conf. Comput. Vision Pattern Recog.*, pages 321–326, 1996.